# Data-Dependent Structural Risk Minimisation for Perceptron Decision Trees

**John Shawe-Taylor**
Dept of Computer Science
Royal Holloway, University of London
Egham, Surrey TW20 0EX, UK
Email: jst@dcs.rhbnc.ac.uk

**Nello Cristianini**
Dept of Engineering Mathematics
University of Bristol
Bristol BS8 1TR, UK
Email: nello.cristianini@bristol.ac.uk

## Abstract

Perceptron Decision Trees (also known as Linear Machine DTs, etc.) are analysed in order that data-dependent Structural Risk Minimization can be applied. Data-dependent analysis is performed which indicates that choosing the maximal margin hyperplanes at the decision nodes will improve the generalization. The analysis uses a novel technique to bound the generalization error in terms of the margins at individual nodes. Experiments performed on real data sets confirm the validity of the approach.

## 1 Introduction

Neural network researchers have traditionally tackled classification problems by assembling perceptron or sigmoid nodes into feedforward neural networks. In this paper we consider a less common approach where the perceptrons are used as decision nodes in a decision tree structure. The approach has the advantage that more efficient heuristic algorithms exist for these structures, while the advantages of inherent parallelism are if anything greater as all the perceptrons can be evaluated in parallel, with the path through the tree determined in a very fast post-processing phase.

Classical Decision Trees (DTs), like the ones produced by popular packages as CART [5] or C4.5 [9], partition the input space by means of axis-parallel hyperplanes (one at each internal node), hence inducing categories which are represented by (axis-parallel) hyperrectangles in such a space.

A natural extension of that hypothesis space is obtained by associating to each internal node hyperplanes in general position, hence partitioning the input space by means of polygonal (polyhedral) categories.

This approach has been pursued by many researchers, often with different motivations, and hence the resulting hypothesis space has been given a number of different names: multivariate DTs [6], oblique DTs [8], or DTs using linear combinations of the attributes [5], Linear Machine DTs, Neural Decision Trees [12], Perceptron Trees [13], etc.

We will call them Perceptron Decision Trees (PDTs), as they can be regarded as binary trees having a simple perceptron associated to each decision node.

Different algorithms for Top-Down induction of PDTs from data have been proposed, based on different principles, [10], [5], [8],

Experimental study of learning by means of PDTs indicates that their performances are sometimes better than those of traditional decision trees in terms of generalization error, and usually much better in terms of tree-size [8], [6], but on some data set PDTs can be outperformed by normal DTs.

We investigate an alternative strategy for improving the generalization of these structures, namely placing maximal margin hyperplanes at the decision nodes. By use of a novel analysis we are able to demonstrate that improved generalization bounds can be obtained for this approach. Experiments confirm that such a method delivers more accurate trees in all tested databases.

## 2  Generalized Decision Trees

**Definition 2.1** *Generalized Decision Trees (GDT).*

Given a space $X$ and a set of boolean functions $\mathcal{F} = \{f : X \to \{0,1\}\}$, the class $\mathrm{GDT}(\mathcal{F})$ of Generalized Decision Trees over $\mathcal{F}$ are functions which can be implemented using a binary tree where each internal node is labeled with an element of $\mathcal{F}$, and each leaf is labeled with either 1 or 0.

To evaluate a particular tree $T$ on input $x \in X$, All the boolean functions associated to the nodes are assigned the same argument $x \in X$, which is the argument of $T(x)$. The values assumed by them determine a unique path from the root to a leaf: at each internal node the left (respectively right) edge to a child is taken if the output of the function associated to that internal node is 0 (respectively 1). The value of the function at the assignment of a $x \in X$ is the value associated to the leaf reached. We say that input $x$ reaches a node of the tree, if that node is on the evaluation path for $x$.

In the following, the *nodes* are the internal nodes of the binary tree, and the *leaves* are its external ones.

**Examples.**

- Given $X = \{0,1\}^n$, a *Boolean Decision Tree (BDT)* is a GDT over
$$\mathcal{F}_{\mathrm{BDT}} = \{f_i : f_i(\mathbf{x}) = \mathbf{x}_i, \forall \mathbf{x} \in X\}$$

- Given $X = \mathbb{R}^n$, a *C4.5-like Decision Tree (CDT)* is a GDT over
$$\mathcal{F}_{\mathrm{CDT}} = \{f_{i,\theta} : f_{i,\theta}(\mathbf{x}) = 1 \Leftrightarrow x_i > \theta\}$$
This kind of decision trees defined on a continuous space are the output of common algorithms like C4.5 and CART, and we will call them - for short - CDTs.

- Given $X = \mathbb{R}^n$, a *Perceptron Decision Tree (PDT)* is a GDT over
$$\mathcal{F}_{\mathrm{PDT}} = \{w^T \mathbf{x} : w \in \mathbb{R}^{n+1}\},$$
where we have assumed that the inputs have been augmented with a coordinate of constant value, hence implementing a thresholded perceptron.

## 3  Data-dependent SRM

We begin with the definition of the fat-shattering dimension, which was first introduced in [7], and has been used for several problems in learning since [1, 4, 2, 3].

**Definition 3.1** *Let $\mathcal{F}$ be a set of real valued functions. We say that a set of points $X$ is $\gamma$-shattered by $\mathcal{F}$ relative to $r = (r_x)_{x \in X}$ if there are real numbers $r_x$ indexed by $x \in X$ such that for all binary vectors $b$ indexed by $X$, there is a function $f_b \in \mathcal{F}$ satisfying*

$$f_b(x) \begin{cases} \geq r_x + \gamma & \text{if } b_x = 1 \\ \leq r_x - \gamma & \text{otherwise.} \end{cases}$$

*The fat shattering dimension $\text{fat}_{\mathcal{F}}$ of the set $\mathcal{F}$ is a function from the positive real numbers to the integers which maps a value $\gamma$ to the size of the largest $\gamma$-shattered set, if this is finite, or infinity otherwise.*

As an example which will be relevant to the subsequent analysis consider the class:

$$\mathcal{F}_{\text{lin}} = \{x \to \langle w, x \rangle + \theta : \|w\| = 1\}.$$

We quote the following result from [11].

**Corollary 3.2** *[11] Let $\mathcal{F}_{\text{lin}}$ be restricted to points in a ball of $n$ dimensions of radius $R$ about the origin and with thresholds $|\theta| \leq R$. Then*

$$\text{fat}_{\mathcal{F}_{\text{lin}}}(\gamma) \leq \min\{9R^2/\gamma^2, n+1\} + 1.$$

The following theorem bounds the generalization of a classifier in terms of the fat shattering dimension rather than the usual Vapnik-Chervonenkis or Pseudo dimension.

Let $T_\theta$ denote the threshold function at $\theta$: $T_\theta : \mathbb{R} \to \{0, 1\}$, $T_\theta(\alpha) = 1$ iff $\alpha > \theta$. For a class of functions $\mathcal{F}$, $T_\theta(\mathcal{F}) = \{T_\theta(f) : f \in \mathcal{F}\}$.

**Theorem 3.3** *[11] Consider a real valued function class $\mathcal{F}$ having fat shattering function bounded above by the function $\text{afat} : \mathbb{R} \to \mathbb{N}$ which is continuous from the right. Fix $\theta \in \mathbb{R}$. If a learner correctly classifies $m$ independently generated examples $\mathbf{z}$ with $h = T_\theta(f) \in T_\theta(\mathcal{F})$ such that $\text{er}_{\mathbf{z}}(h) = 0$ and $\gamma = \min|f(x_i) - \theta|$, then with confidence $1 - \delta$ the expected error of $h$ is bounded from above by*

$$\epsilon(m, k, \delta) = \frac{2}{m}\left(k \log\left(\frac{8em}{k}\right)\log(32m) + \log\left(\frac{8m}{\delta}\right)\right),$$

*where $k = \text{afat}(\gamma/8)$.*

The importance of this theorem is that it can be used to explain how a classifier can give better generalization than would be predicted by a classical analysis of its VC dimension. Essentially expanding the margin performs an automatic capacity control for function classes with small fat shattering dimensions. The theorem shows that when a large margin is achieved it is as if we were working in a lower VC class.

We should stress that in general the bounds obtained should be better for cases where a large margin is observed, but that a priori there is no guarantee that such a margin will occur. Therefore a priori only the classical VC bound can be used. In view of corresponding lower bounds on the generalization error in terms of the VC dimension, the a posteriori bounds depend on a favourable probability distribution making the actual learning task easier. Hence, the result will only be useful if the distribution is favourable or at least not adversarial. In this sense the result is a distribution dependent result, despite not being distribution dependent in the

traditional sense that assumptions about the distribution have had to be made in its derivation. The benign behaviour of the distribution is automatically estimated in the learning process.

In order to perform a similar analysis for perceptron decision trees we will consider the set of margins obtained at each of the nodes, bounding the generalization as a function of these values.

## 4  Generalisation analysis of the Tree Class

It turns out that bounding the fat shattering dimension of PDT's viewed as real function classifiers is difficult. We will therefore do a direct generalization analysis mimicking the proof of Theorem 3.3 but taking into account the margins at each of the decision nodes in the tree.

**Definition 4.1** *Let $(X, d)$ be a (pseudo-) metric space, let $A$ be a subset of $X$ and $\epsilon > 0$. A set $B \subseteq X$ is an $\epsilon$-cover for $A$ if, for every $a \in A$, there exists $b \in B$ such that $d(a, b) < \epsilon$. The $\epsilon$-covering number of $A$, $\mathcal{N}_d(\epsilon, A)$, is the minimal cardinality of an $\epsilon$-cover for $A$ (if there is no such finite cover then it is defined to be $\infty$).*

We write $\mathcal{N}(\epsilon, \mathcal{F}, \mathbf{x})$ for the $\epsilon$-covering number of $\mathcal{F}$ with respect to the $\ell_\infty$ pseudo-metric measuring the maximum discrepancy on the sample $\mathbf{x}$. These numbers are bounded in the following Lemma.

**Lemma 4.2 (Alon** *et al.* **[1])** *Let $\mathcal{F}$ be a class of functions $X \to [0, 1]$ and $P$ a distribution over $X$. Choose $0 < \epsilon < 1$ and let $d = \mathrm{fat}_{\mathcal{F}}(\epsilon/4)$. Then*

$$E\left(\mathcal{N}(\epsilon, \mathcal{F}, \mathbf{x})\right) \le 2\left(\frac{4m}{\epsilon^2}\right)^{d\log(2em/(d\epsilon))},$$

*where the expectation $E$ is taken w.r.t. a sample $\mathbf{x} \in X^m$ drawn according to $P^m$.*

**Corollary 4.3** *[11] Let $\mathcal{F}$ be a class of functions $X \to [a, b]$ and $P$ a distribution over $X$. Choose $0 < \epsilon < 1$ and let $d = \mathrm{fat}_{\mathcal{F}}(\epsilon/4)$. Then*

$$E\left(\mathcal{N}(\epsilon, \mathcal{F}, \mathbf{x})\right) \le 2\left(\frac{4m(b-a)^2}{\epsilon^2}\right)^{d\log(2em(b-a)/(d\epsilon))},$$

*where the expectation $E$ is over samples $\mathbf{x} \in X^m$ drawn according to $P^m$.*

We are now in a position to tackle the main lemma which bounds the probability over a double sample that the first half has zero error and the second error greater than an appropriate $\epsilon$. Here, error is interpreted as being differently classified at the output of tree. In order to simplify the notation in the following lemma we assume that the decision tree has $K$ nodes. We also denote $\mathrm{fat}_{\mathcal{F}_{\mathrm{lin}}}(\gamma)$ by $\mathrm{fat}(\gamma)$ to simplify the notation.

**Lemma 4.4** *Let $T$ be a perceptron decision tree with $K$ decision nodes with margins $\gamma^1, \gamma^2, \ldots, \gamma^K$ at the decision nodes. If it has correctly classified $m$ labelled examples generated independently according to the unknown (but fixed) distribution $P$, then we can bound the following probability to be less than $\delta$,*

$$P^{2m}\left\{\mathbf{xy}: \exists \text{ a tree } T: T \text{ correctly classifies } \mathbf{x},\right.$$

$$\left. \text{fraction of } \mathbf{y} \text{ misclassified} > \epsilon(m, K, \delta) \right\} < \delta,$$

*where $\epsilon(m, K, \delta) = \frac{1}{m}\left(D\log(4m) + \log\frac{2^K}{\delta}\right)$.*
*where $D = \sum_{i=1}^{K} k_i \log(4em/k_i)$ and $k_i = \mathrm{fat}(\gamma_i/8)$.*

**Proof**: Using the standard permutation argument, we may fix a sequence **xy** and bound the probability under the uniform distribution on swapping permutations that the sequence satisfies the condition stated. We consider generating minimal $\gamma_k/2$-covers $B^k_{\mathbf{xy}}$ for each value of $k$, where $\gamma_k = \min\{\gamma' : \mathrm{fat}(\gamma'/8) \le k\}$. Suppose that for node $i$ of the tree the margin $\gamma^i$ of the hyperplane $w_i$ satisfies $\mathrm{fat}(\gamma^i/8) = k_i$. We can therefore find $f_i \in B^{k_i}_{\mathbf{xy}}$ whose output values are within $\gamma^i/2$ of $w_i$. We now consider the tree $T'$ obtained by replacing the node perceptrons $w_i$ of $T$ with the corresponding $f_i$. This tree performs the same classification function on the first half of the sample, and the margin remains larger than $\gamma^i - \gamma_{k_i}/2 > \gamma_{k_i}/2$. If a point in the second half of the sample is incorrectly classified by $T$ it will either still be incorrectly classified by the adapted tree $T'$ or will at one of the decision nodes $i$ in $T'$ be closer to the decision boundary than $\gamma_{k_i}/2$. The point is thus distinguishable from left hand side points which are both correctly classified and have margin greater than $\gamma_{k_i}/2$ at node $i$. Hence, that point must be kept on the right hand side in order for the condition to be satisfied. Hence, the fraction of permutations that can be allowed for one choice of the functions from the covers is $2^{-\epsilon m}$. We must take the union bound over all choices of the functions from the covers. Using the techniques of [11] the numbers of these choices is bounded by Corollory 4.3 as follows

$$\Pi^K_{i=1} 2(8m)^{k_i \log(4em/k_i)} = 2^K (8m)^D,$$

where $D = \sum^K_{i=1} k_i \log(4em/k_i)$. The value of $\epsilon$ in the lemma statement therefore ensures that this the union bound is less than $\delta$.

□

Using the standard lemma due to Vapnik [14, page 168] to bound the error probabilities in terms of the discrepancy on a double sample, combined with Lemma 4.4 gives the following result.

**Theorem 4.5** *Suppose we are able to classify an $m$ sample of labelled examples using a perceptron decision tree with $K$ nodes and obtaining margins $\gamma_i$ at node $i$, then we can bound the generalisation error with probability greater than $1 - \delta$ to be less than*

$$\frac{1}{m}\left(D\log(4m) + \log\frac{(8m)^K \binom{2K}{K}}{(K+1)\delta}\right)$$

*where $D = \sum^K_{i=1} k_i \log(4em/k_i)$ and $k_i = \mathrm{fat}(\gamma_i/8)$.*

**Proof**: We must bound the probabilities over different architectures of trees and different margins. We simply have to choose the values of $\epsilon$ to ensure that the individual $\delta$'s are sufficiently small that the total over all possible choices is less than $\delta$. The details are omitted in this abstract.

□

## 5  Experiments

The theoretical results obtained in the previous section imply that an algorithm which produces large margin splits should have a better generalization, since increasing the margins in the internal nodes, has the effect of decreasing the bound on the test error.

In order to test this strategy, we have performed the following experiment, divided in two parts: first run a standard perceptron decision tree algorithm and then for each decision node generate a maximal margin hyperplane implementing the same dichotomy in place of the decision boundary generated by the algorithm.

**Input:** Random $m$ sample $\mathbf{x}$ with corresponding classification $b$.

**Algorithm:** Find a perceptron decision tree $T$ which correctly classifies the sample using a standard algorithm;
Let $k = $ number of decision nodes of $T$;
From tree $T$ create $T'$ by executing the following loop:

    **For each decision node** $i$ replace the weight vector $w_i$ by the vector $w_i'$ which realises the maximal margin hyperplane agreeing with $w_i$ on the set of inputs reaching node $i$;

    Let the margin of $w_i'$ on the inputs reaching node $i$ be $\gamma_i$;

**Output:** Classifier $T'$, with bound on the generalisation error in terms of the number of decision nodes $K$ and $D = \sum_{i=1}^{K} k_i \log(4em/k_i)$ where $k_i = \text{fat}(\gamma_i/8)$.

Note that the classification of $T$ and $T'$ agree on the sample and hence, that $T'$ is consistent with the sample.

As a PDT learning algorithm we have used OC1 [8], created by Murthy, Kasif and Salzberg and freely available over the internet. It is a randomized algorithm, which performs simulated annealing for learning the perceptrons. The details about the randomization, the pruning, and the splitting criteria can be found in [8].

The data we have used for the test are 4 of the 5 sets used in the original OC1 paper, which are publicly available in the UCI data repository [16].

The results we have obtained on these data are compatible with the ones reported in the original OC1 paper, the differences being due to different divisions between training and testing sets and their sizes; the absence in our experiments of cross-validation and other techniques to estimate the predictive accuracy of the PDT; and the inherently randomized nature of the algorithm.

The second stage of the experiment involved finding - for each node - the hyperplane which performes *the same* split as performed by the OC1 tree but with the maximal margin. This can be done by considering the subsample reaching each node as perfectly divided in two parts, and feeding the data accordingly relabelled to an algorithm which finds the optimal split in the linearly separable case. The maximal margin hyperplanes are then placed in the decision nodes and the new tree is tested on the same testing set.

The data sets we have used are: *Wiscounsin Breast Cancer*, *Pima Indians Diabetes*, *Boston Housing* transformed into a classification problem by thresholding the price at $ 21.000 and the classical *Iris* studied by Fisher (More informations about the databases and their authors are in [8]). All the details about sample sizes, number of attributes and results (training and testing accuracy, tree size) are summarized in table 1.

We were not particularly interested in achieving a high testing accuracy, but rather in observing if improved performances can be obtained by increasing the margin. For this reason we did not try to optimize the performance of the original classifier by using cross-validation, or a convenient training/testing set ratio. The relevant quantity, in this experiment, is the different in the testing error between a PDT with arbitrary margins and the same tree with optimized margins. This quantity has turned out to be always positive, and to range from 1.7 to 2.8 percent of gain, on test errors which were already very low.

|      | train | OC1 test | FAT test | #trs | #ts | attrib. | classes | nodes |
|------|-------|----------|----------|------|-----|---------|---------|-------|
| CANC | 96.53 | 93.52    | 95.37    | 249  | 108 | 9       | 2       | 1     |
| IRIS | 96.67 | 96.67    | 98.33    | 90   | 60  | 4       | 3       | 2     |
| DIAB | 89.00 | 70.48    | 72.45    | 209  | 559 | 8       | 2       | 4     |
| HOUS | 95.90 | 81.43    | 84.29    | 306  | 140 | 13      | 2       | 7     |

# References

[1] Ncga Alon, Shai Ben-David, Nicolò Cesa-Bianchi and David Haussler, "Scale-sensitive Dimensions, Uniform Convergence, and Learnability," in *Proceedings of the Conference on Foundations of Computer Science (FOCS)*, (1993). Also to appear in *Journal of the ACM*.

[2] Martin Anthony and Peter Bartlett, "Function learning from interpolation", Technical Report, (1994). (An extended abstract appeared in *Computational Learning Theory, Proceedings 2nd European Conference, EuroCOLT'95*, pages 211–221, ed. Paul Vitanyi, (Lecture Notes in Artificial Intelligence, 904) Springer-Verlag, Berlin, 1995).

[3] Peter L. Bartlett and Philip M. Long, "Prediction, Learning, Uniform Convergence, and Scale-Sensitive Dimensions," Preprint, Department of Systems Engineering, Australian National University, November 1995.

[4] Peter L. Bartlett, Philip M. Long, and Robert C. Williamson, "Fat-shattering and the learnability of Real-valued Functions," *Journal of Computer and System Sciences*, 52(3), 434-452, (1996).

[5] Breiman L., Friedman J.H., Olshen R.A., Stone C.J., "Classification and Regression Trees", Wadsworth International Group, Belmont, CA, 1984.

[6] Brodley C.E., Utgoff P.E., Multivariate Decision Trees, Machine Learning 19, pp. 45-77, 1995.

[7] Michael J. Kearns and Robert E. Schapire, "Efficient Distribution-free Learning of Probabilistic Concepts," pages 382–391 in *Proceedings of the 31st Symposium on the Foundations of Computer Science*, IEEE Computer Society Press, Los Alamitos, CA, 1990.

[8] Murthy S.K., Kasif S., Salzberg S., A System for Induction of Oblique Decision Trees, Journal of Artificial Intelligence Research, 2 (1994), pp. 1-32.

[9] Quinlan J.R., "C4.5: Programs for Machine Learning", Morgan Kaufmann, 1993.

[10] Sankar A., Mammone R.J., Growing and Pruning Neural Tree Networks, IEEE Transactions on Computers, 42:291-299, 1993.

[11] John Shawe-Taylor, Peter L. Bartlett, Robert C. Williamson, Martin Anthony, Structural Risk Minimization over Data-Dependent Hierarchies, NeuroCOLT Technical Report NC-TR-96-053, 1996.
(ftp://ftp.dcs.rhbnc.ac.uk/pub/neurocolt/tech_reports).

[12] J.A. Sirat, and J.-P. Nadal, "Neural trees: a new tool for classification", Network, 1, pp. 423-438, 1990

[13] Utgoff P.E., Perceptron Trees: a Case Study in Hybrid Concept Representations, Connection Science 1 (1989), pp. 377-391.

[14] Vladimir N. Vapnik, *Estimation of Dependences Based on Empirical Data*, Springer-Verlag, New York, 1982.

[15] Vladimir N. Vapnik, *The Nature of Statistical Learning Theory*, Springer-Verlag, New York, 1995

[16] University of California, Irvine - Machine Learning Repository, http://www.ics.uci.edu/ mlearn/MLRepository.html
